# Feature Selection and Classification on Matrix Data: From Large Margins To Small Covering Numbers

**Sepp Hochreiter and Klaus Obermayer**
Department of Electrical Engineering and Computer Science
Technische Universität Berlin
10587 Berlin, Germany
{hochreit,oby}@cs.tu-berlin.de

## Abstract

We investigate the problem of learning a classification task for datasets which are described by matrices. Rows and columns of these matrices correspond to objects, where row and column objects may belong to different sets, and the entries in the matrix express the relationships between them. We interpret the matrix elements as being produced by an unknown kernel which operates on object pairs and we show that - under mild assumptions - these kernels correspond to dot products in some (unknown) feature space. Minimizing a bound for the generalization error of a linear classifier which has been obtained using covering numbers we derive an objective function for model selection according to the principle of structural risk minimization. The new objective function has the advantage that it allows the analysis of matrices which are not positive definite, and not even symmetric or square. We then consider the case that row objects are interpreted as features. We suggest an additional constraint, which imposes sparseness on the row objects and show, that the method can then be used for feature selection. Finally, we apply this method to data obtained from DNA microarrays, where "column" objects correspond to samples, "row" objects correspond to genes and matrix elements correspond to expression levels. Benchmarks are conducted using standard one-gene classification and support vector machines and K-nearest neighbors after standard feature selection. Our new method extracts a sparse set of genes and provides superior classification results.

## 1  Introduction

Many properties of sets of objects can be described by matrices, whose rows and columns correspond to objects and whose elements describe the relationship between them. One typical case are so-called pairwise data, where rows as well as columns of the matrix represent the objects of the dataset (Fig. 1a) and where the entries of the matrix denote similarity values which express the relationships between objects.

# Pairwise Data (a)  Feature Vectors (b)

|   | A | B | C | D | E | F | G | H | I | J | K | L |
|---|---|---|---|---|---|---|---|---|---|---|---|---|
| A | 0.9 | -0.1 | -0.8 | 0.5 | 0.2 | -0.5 | -0.7 | -0.9 | 0.2 | -0.7 | 0.4 | -0.3 |
| B | -0.1 | 0.9 | 0.6 | 0.3 | -0.7 | -0.6 | 0.3 | 0.7 | -0.3 | -0.8 | -0.7 | -0.9 |
| C | -0.8 | 0.6 | 0.9 | 0.2 | -0.6 | 0.6 | 0.5 | 0.2 | -0.7 | -0.5 | -0.1 | 0.6 |
| D | 0.5 | 0.3 | 0.2 | 0.9 | 0.7 | 0.1 | 0.3 | -0.1 | 0.6 | 0.9 | -0.9 | -0.1 |
| E | 0.2 | -0.7 | -0.6 | 0.7 | 0.9 | -0.9 | -0.5 | 0.4 | 0.1 | 0.3 | -0.6 | 0.7 |
| F | -0.5 | -0.6 | 0.6 | 0.1 | -0.9 | 0.9 | 0.9 | -0.2 | -0.6 | -0.5 | -0.4 | -0.3 |
| G | -0.7 | 0.3 | 0.5 | 0.3 | -0.5 | 0.9 | 0.9 | -0.3 | -0.3 | 0.6 | 0.9 | -0.7 |
| H | -0.9 | 0.7 | 0.2 | -0.1 | 0.4 | -0.2 | -0.3 | 0.9 | 0.2 | -0.9 | 0.3 | 0.4 |
| I | 0.2 | -0.3 | -0.7 | 0.6 | 0.1 | -0.6 | -0.3 | 0.2 | 0.9 | -0.3 | -0.7 | 0.8 |
| J | -0.7 | -0.8 | -0.5 | 0.9 | -0.3 | -0.5 | 0.6 | -0.9 | -0.3 | 0.9 | -0.1 | -0.5 |
| K | 0.4 | -0.7 | -0.1 | -0.9 | -0.6 | -0.4 | 0.9 | 0.3 | -0.7 | -0.1 | 0.9 | 0.1 |
| L | -0.3 | -0.9 | 0.6 | -0.1 | 0.7 | -0.3 | -0.7 | 0.4 | 0.8 | -0.5 | 0.1 | 0.9 |

|   | A | B | C | D | E | F | G |
|---|---|---|---|---|---|---|---|
| $\alpha$ | 1.3 | -2.2 | -1.6 | 7.8 | 6.6 | -7.5 | -4.8 |
| $\beta$ | -1.8 | -1.1 | 7.2 | 2.3 | 9.0 | 3.8 | 3.9 |
| $\chi$ | 1.2 | 1.9 | -2.9 | -2.2 | -4.4 | -4.7 | -8.4 |
| $\delta$ | 3.7 | 0.8 | -0.6 | 2.5 | -5.7 | 0.1 | -0.3 |
| $\varepsilon$ | 9.2 | -9.4 | -8.3 | 9.2 | -2.4 | -3.9 | 1.9 |
| $\phi$ | -7.7 | 8.6 | -9.7 | -7.4 | 2.6 | 6.9 | 2.9 |
| $\gamma$ | -4.8 | 0.1 | -1.2 | 0.9 | 0.2 | 2.7 | 0.2 |
| $\eta$ | 0.7 | -1.7 | 0.3 | -7.2 | -1.8 | 4.6 | 2.6 |
| $\iota$ | -6.2 | -6.2 | 1.8 | 3.6 | -0.7 | -9.4 | 0.9 |
| $\varphi$ | 9.0 | 4.8 | -8.3 | -0.8 | -2.0 | 4.4 | -1.9 |
| $\kappa$ | 6.2 | 9.0 | 1.5 | -1.1 | 7.7 | 8.4 | -2.1 |
| $\lambda$ | 9.6 | 7.0 | 2.5 | -4.3 | -5.4 | 0.7 | 1.2 |

Figure 1: Two typical examples of matrix data (see text). (a) Pairwise data. Row (A-L) and column (A-L) objects coincide. (b) Feature vectors. Column objects (A-G) differ form row objects ($\alpha$ - $\lambda$). The latter are interpreted as features.

Another typical case occurs, if objects are described by a set of features (Fig. 1b). In this case, the column objects are the objects to be characterized, the row objects correspond to their features and the matrix elements denote the strength with which a feature is expressed in a particular object.

In the following we consider the task of learning a classification problem on matrix data. We consider the case that class labels are assigned to the column objects of the training set. Given the matrix and the class labels we then want to construct a classifier with good generalization properties. From all the possible choices we select classifiers from the support vector machine (SVM) family [1, 2] and we use the principle of structural risk minimization [15] for model selection - because of its recent success [11] and its theoretical properties [15].

Previous work on large margin classifiers for datasets, where objects are described by feature vectors and where SVMs operate on the column vectors of the matrix, is abundant. However, there is one serious problem which arise when the number of features becomes large and comparable to the number of objects: Without feature selection, SVMs are prone to overfitting, despite the complexity regularization which is implicit in the learning method [3]. Rather than being sparse in the number of support vectors, the classifier should be sparse in the number of features used for classification. This relates to the result [15] that the number of features provide an upper bound on the number of "essential" support vectors.

Previous work on large margin classifiers for datasets, where objects are described by their mutual similarities, was centered around the idea that the matrix of similarities can be interpreted as a Gram matrix (see e.g. Hochreiter & Obermayer [7]). Work along this line, however, was so far restricted to the case (i) that the Gram matrix is positive definite (although methods have been suggested to modify indefinite Gram matrices in order to restore positive definiteness [10]) and (ii) that row and column objects are from the same set (pairwise data) [7].

In this contribution we extend the Gram matrix approach to matrix data, where row and column objects belong to different sets. Since we can no longer expect that the matrices are positive definite (or even square), a new objective function must be derived. This is done in the next section, where an algorithm for the construction of linear classifiers is derived using the principle of structural risk minimization. Section 3 is concerned with the question under what conditions matrix elements can indeed be interpreted as vector products in some feature space. The method is specialized to pairwise data in Section 4. A sparseness constraint for feature selection is introduced in Section 5. Section 6, finally, contains an evaluation of the new method for DNA microarray data as well as benchmark results with standard classifiers which are based on standard feature selection procedures.

## 2  Large Margin Classifiers for Matrix Data

In the following we consider two sets $\mathcal{X}$ and $\mathcal{Z}$ of objects, which are described by feature vectors $\boldsymbol{x}$ and $\boldsymbol{z}$. Based on the feature vectors $\boldsymbol{x}$ we construct a linear classifier defined through the classification function

$$f(\boldsymbol{x}) \;=\; \langle \boldsymbol{w}, \boldsymbol{x} \rangle \;+\; b, \tag{1}$$

where $\langle .,. \rangle$ denotes a dot product. The zero isoline of $f$ is a hyperplane which is parameterized by its unit normal vector $\hat{\boldsymbol{w}}$ and by its perpendicular distance $b/\|\boldsymbol{w}\|_2$ from the origin. The hyperplane's margin $\gamma$ with respect to $\mathcal{X}$ is given by

$$\gamma \;=\; \min_{\boldsymbol{x} \in \mathcal{X}} |\langle \hat{\boldsymbol{w}}, \boldsymbol{x} \rangle \;+\; b/\|\boldsymbol{w}\|_2 \,|. \tag{2}$$

Setting $\gamma \;=\; \|\boldsymbol{w}\|_2^{-1}$ allows us to treat normal vectors $\boldsymbol{w}$ which are not normalized, if the margin is normalized to 1. According to [15] this is called the "canonical form" of the separation hyperplane. The hyperplane with largest margin is then obtained by minimizing $\|\boldsymbol{w}\|_2^2$ for a margin which equals 1.

It has been shown [14, 13, 12] that the generalization error of a linear classifier, eq. (1), can be bounded from above with probability $1 - \delta$ by the bound $\mathcal{B}$,

$$\mathcal{B}(L, a/\gamma, \delta) \;=\; \frac{2}{L}\left(\log_2\left(E\mathcal{N}\left(\frac{\gamma}{2\,a}, \mathcal{F}, 2L\right)\right) \;+\; \log_2\left(\frac{4\,L\,a}{\delta\,\gamma}\right)\right), \tag{3}$$

provided that the training classification error is zero and $f(\boldsymbol{x})$ is bounded by $-a \le f(\boldsymbol{x}) \le a$ for all $\boldsymbol{x}$ drawn iid from the (unknown) distribution of objects. $L$ denotes the number of training objects $\boldsymbol{x}$, $\gamma$ denotes the margin and $E\mathcal{N}(\epsilon, \mathcal{F}, L)$ the expected $\epsilon$-covering number of a class $\mathcal{F}$ of functions that map data objects from $T$ to $[0, 1]$ (see Theorem 7.7 in [14] and Proposition 19 in [12]). In order to obtain a classifier with good generalization properties we suggest to minimize $a/\gamma$ under proper constraints. $a$ is not known in general, however, because the probability distribution of objects (in particular its support) is not known. In order to avoid this problem we approximate $a$ by the range $m \;=\; 0.5\,\left(\max_i\langle \hat{\boldsymbol{w}}, \boldsymbol{x}^i \rangle - min_i\langle \hat{\boldsymbol{w}}, \boldsymbol{x}^i \rangle\right)$ of values in the training set and minimize the quantity $\mathcal{B}(L, m/\gamma, \delta)$ instead of eq. (3).

Let $\boldsymbol{X} := \left(\boldsymbol{x}^1, \boldsymbol{x}^2, \dots, \boldsymbol{x}^L\right)$ be the matrix of feature vectors of $L$ objects from the set $\mathcal{X}$ and $\boldsymbol{Z} := \left(\boldsymbol{z}^1, \boldsymbol{z}^2, \dots, \boldsymbol{z}^P\right)$ be the matrix of feature vectors of $P$ objects from the set $\mathcal{Z}$. The objects of set $\mathcal{X}$ are labeled, and we summarize all labels using a label matrix $\boldsymbol{Y} : [\boldsymbol{Y}]_{ij} := y^i \delta_{ij} \in \mathbb{R}^{L \times L}$, where $\delta$ is the Kronecker-Delta. Let us consider the case that the feature vectors $\boldsymbol{X}$ and $\boldsymbol{Z}$ are unknown, but that we are given the matrix $\boldsymbol{K} := \boldsymbol{X}^T \boldsymbol{Z}$ of the corresponding scalar products. The training set is then given by the data matrix $\boldsymbol{K}$ and the corresponding label matrix $\boldsymbol{Y}$. The principle of structural risk minimization is implemented by minimizing an upper bound on

$(m/\gamma)^2$ given by $\|\boldsymbol{X}^T \boldsymbol{w}\|_2^2$, as can be seen from $m/\gamma \leq \|\boldsymbol{w}\|_2 \max_i |\langle \hat{\boldsymbol{w}}, \boldsymbol{x}^i \rangle| \leq \sqrt{\sum_i (\langle \boldsymbol{w}, \boldsymbol{x}^i \rangle)^2} = \|\boldsymbol{X}^T \boldsymbol{w}\|_2$. The constraints $f(\boldsymbol{x}^i) = y^i$ imposed by the training set are taken into account using the expressions $1 - \xi_i^+ \leq y^i (\langle \boldsymbol{w}, \boldsymbol{x}^i \rangle + b) \leq 1 + \xi_i^-$, where $\xi_i^+, \xi_i^- \geq 0$ are slack variables which should also be minimized. We thus obtain the optimization problem

$$\min_{\boldsymbol{w}, b, \boldsymbol{\xi}^+, \boldsymbol{\xi}^-} \quad \frac{1}{2} \|\boldsymbol{X}^T \boldsymbol{w}\|_2^2 + M^+ \mathbf{1}^T \boldsymbol{\xi}^+ + M^- \mathbf{1}^T \boldsymbol{\xi}^- \tag{4}$$

$$\text{s.t.} \quad \boldsymbol{Y}^{-1} \left(\boldsymbol{X}^T \boldsymbol{w} + b\mathbf{1}\right) - \mathbf{1} + \boldsymbol{\xi}^+ \geq \mathbf{0}$$

$$\boldsymbol{Y}^{-1} \left(\boldsymbol{X}^T \boldsymbol{w} + b\mathbf{1}\right) - \mathbf{1} - \boldsymbol{\xi}^- \leq \mathbf{0}$$

$$\boldsymbol{\xi}^+, \boldsymbol{\xi}^- \geq \mathbf{0} \ .$$

$M^+$ penalizes wrong classification and $M^-$ absolute values exceeding 1. For classification $M^-$ may be set to zero. Note, that the quadratic expression in the objective function is convex, which follows from $\|\boldsymbol{X}^T \boldsymbol{w}\|_2^2 = \boldsymbol{w}^T \boldsymbol{X} \boldsymbol{X}^T \boldsymbol{w}$ and the fact that $\boldsymbol{X} \boldsymbol{X}^T$ is positive semidefinite.

Let $\tilde{\boldsymbol{\alpha}}^+, \tilde{\boldsymbol{\alpha}}^-$ be the dual variables for the constraints imposed by the training set, $\tilde{\boldsymbol{\alpha}} := \tilde{\boldsymbol{\alpha}}^+ - \tilde{\boldsymbol{\alpha}}^-$, and $\boldsymbol{\alpha}$ a vector with $\tilde{\boldsymbol{\alpha}} = \boldsymbol{Y} \left(\boldsymbol{X}^T \boldsymbol{Z}\right) \boldsymbol{\alpha}$. Two cases must be treated: $\boldsymbol{\alpha}$ is not unique or does not exist. First, if $\boldsymbol{\alpha}$ is not unique we choose $\boldsymbol{\alpha}$ according to Section 5. Second, if $\boldsymbol{\alpha}$ does not exist we set $\boldsymbol{\alpha} = \left(\boldsymbol{Z}^T \boldsymbol{X} \boldsymbol{Y}^{-T} \boldsymbol{Y}^{-1} \boldsymbol{X}^T \boldsymbol{Z}\right)^{-1} \boldsymbol{Z}^T \boldsymbol{X} \boldsymbol{Y}^{-T} \tilde{\boldsymbol{\alpha}}$, where $\boldsymbol{Y}^{-T} \boldsymbol{Y}^{-1}$ is the identity. The optimality conditions require that the following derivatives of the Lagrangian $L$ are zero: $\partial L/\partial b = \mathbf{1}^T \boldsymbol{Y}^{-1} \tilde{\boldsymbol{\alpha}}$, $\partial L/\partial \boldsymbol{w} = \boldsymbol{X} \boldsymbol{X}^T \boldsymbol{w} - \boldsymbol{X} \boldsymbol{Y}^{-1} \tilde{\boldsymbol{\alpha}}$, $\partial L/\partial \boldsymbol{\xi}^\pm = M^\pm \mathbf{1} - \tilde{\boldsymbol{\alpha}}^\pm + \boldsymbol{\mu}^\pm$, where $\boldsymbol{\mu}^+, \boldsymbol{\mu}^- \geq 0$ are the Lagrange multipliers for the slack variables. We obtain $\boldsymbol{Z}^T \boldsymbol{X} \boldsymbol{X}^T (\boldsymbol{w} - \boldsymbol{Z} \boldsymbol{\alpha}) = 0$ which is ensured by $\boldsymbol{w} = \boldsymbol{Z} \boldsymbol{\alpha}$, $0 = \mathbf{1}^T \left(\boldsymbol{X}^T \boldsymbol{Z}\right) \boldsymbol{\alpha}$, $\tilde{\alpha}_i \leq M^+$, and $-\tilde{\alpha}_i \leq M^-$. The Karush–Kuhn–Tucker conditions give $b = \left(\mathbf{1}^T \boldsymbol{Y} \ \mathbf{1}\right) / \left(\mathbf{1}^T \mathbf{1}\right)$ if $\tilde{\alpha}_i < M^+$ and $-\tilde{\alpha}_i < M^-$.

In the following we set $M^+ = M^- = M$ and $C := M \|\boldsymbol{Y} \left(\boldsymbol{X}^T \boldsymbol{Z}\right)\|_{row}^{-1}$ so that $\|\boldsymbol{\alpha}\|_\infty \leq C$ implies $\|\tilde{\boldsymbol{\alpha}}\|_\infty \leq \|\boldsymbol{Y} \left(\boldsymbol{X}^T \boldsymbol{Z}\right)\|_{row} \|\boldsymbol{\alpha}\|_\infty \leq M$, where $\|.\|_{row}$ is the row-sum norm. We then obtain the following dual problem of eq. (4):

$$\min_{\boldsymbol{\alpha}} \quad \frac{1}{2}\boldsymbol{\alpha}^T \boldsymbol{K}^T \boldsymbol{K} \boldsymbol{\alpha} - \mathbf{1}^T \boldsymbol{Y} \boldsymbol{K} \boldsymbol{\alpha} \tag{5}$$

$$\text{subject to} \quad \mathbf{1}^T \boldsymbol{K} \boldsymbol{\alpha} = 0 \ , \ |\alpha_i| \leq C.$$

If $M^+ \neq M^-$ we must add another constraint. For $M^- = 0$, for example, we have to add $\boldsymbol{Y} \boldsymbol{K} \left(\boldsymbol{\alpha}^+ - \boldsymbol{\alpha}^-\right) \geq \mathbf{0}$. If a classifier has been selected according to eq. (5), a new example $\boldsymbol{u}$ is classified according to the sign of

$$f(\boldsymbol{u}) = \langle \boldsymbol{w}, \boldsymbol{u} \rangle + b = \sum_{i=1}^{P} \alpha_i \langle \boldsymbol{z}^i, \boldsymbol{u} \rangle + b. \tag{6}$$

The optimal classifier is selected by optimizing eq. (5), and as long as $a = m$ holds true for all possible objects $\boldsymbol{x}$ (which are assumed to be drawn iid), the generalization error is bounded by eq. (3). If outliers are rejected, condition $a = m$ can always be enforced. For large training sets the number of rejections is small: The probability $P\{|\langle \boldsymbol{w}, \boldsymbol{x} \rangle| > m\}$ that an outlier occurs can be bounded with confidence $1 - \delta$ using the additive Chernoff bounds (e.g. [15]):

$$P\{|\langle \boldsymbol{w}, \boldsymbol{x} \rangle| > m\} \leq \sqrt{\frac{-\log \delta}{2L}} \ . \tag{7}$$

But note, that not all outliers are misclassified, and the trivial bound on the generalization error is still of the order $L^{-1}$.

# 3 Kernel Functions, Measurements and Scalar Products

In the last section we have assumed that the matrix $\boldsymbol{K}$ is derived from scalar products between the feature vectors $\boldsymbol{x}$ and $\boldsymbol{z}$ which describe the objects from the sets $\mathcal{X}$ and $\mathcal{Z}$. For all practical purposes, however, the only information available is summarized in the matrices $\boldsymbol{K}$ and $\boldsymbol{Y}$. The feature vectors are not known and it is even unclear whether they exist. In order to apply the results of Section 2 to practical problems the following question remains to be answered: What are the conditions under which the measurement operator $k(.,\boldsymbol{z})$ can indeed be interpreted as a scalar product between feature vectors and under which the matrix $\boldsymbol{K}$ can be interpreted as a matrix of kernel evaluations?

In order to answer these questions, we make use of the following theorems. Let $L^2(H)$ denote the set of functions $h$ from $H$ with $\int h^2(\boldsymbol{x})d\boldsymbol{x} < \infty$ and $\ell^2$ the set of infinite vectors $(a_1, a_2, \dots)$ where $\sum_i a_i^2$ converges.

**Theorem 1 (Singular Value Expansion)** *Let $H_1$ and $H_2$ be Hilbert spaces. Let $\alpha$ be from $L^2(H_1)$ and let $k$ be a kernel from $L^2(H_2, H_1)$ which defines a Hilbert-Schmidt operator $T_k : H_1 \to H_2$*

$$(T_k\alpha)(\boldsymbol{x}) = f(\boldsymbol{x}) = \int k(\boldsymbol{x}, \boldsymbol{z}) \ \alpha(\boldsymbol{z}) \ d\boldsymbol{z} \ . \tag{8}$$

*Then there exists an expansion $k(\boldsymbol{x}, \boldsymbol{z}) = \sum_n s_n \ e_n(\boldsymbol{z}) \ g_n(\boldsymbol{x})$ which converges in the $L^2$-sense. The $s_n \geq 0$ are the singular values of $T_k$, and $e_n \in H_1$, $g_n \in H_2$ are the corresponding orthonormal functions.*

**Corollary 1 (Linear Classification in $\ell^2$)** *Let the assumptions of Theorem 1 hold and let $\int_{H_1}(k(\boldsymbol{x}, \boldsymbol{z}))^2 \ d\boldsymbol{z} \leq K^2$ for all $\boldsymbol{x}$. Let $\langle.\rangle_{H_1}$ be the a dot product in $H_1$. We define $\boldsymbol{w} := (\langle\alpha, e_1\rangle_{H_1}, \langle\alpha, e_2\rangle_{H_1}, \dots)$, and $\phi(\boldsymbol{x}) := (s_1 g_1(\boldsymbol{x}), s_2 g_2(\boldsymbol{x}), \dots)$.*

*Then the following holds true:*

- *$\boldsymbol{w}, \phi(\boldsymbol{x}) \in \ell^2$, where $\|w\|_{\ell^2}^2 = \|\alpha\|_{H_1}^2$, and*

- *$\|f\|_{H_2}^2 = \langle T_k^* T_k \alpha, \alpha\rangle_{H_1}$, where $T_k^*$ is the adjoint operator of $T_k$,*

*and the following sum convergences absolutely and uniformly:*

$$f(\boldsymbol{x}) = \langle\boldsymbol{w}, \phi(\boldsymbol{x})\rangle_{\ell^2} = \sum_n s_n \ \langle\alpha, e_n\rangle_{H_1} \ g_n(\boldsymbol{x}) \ . \tag{9}$$

Eq. (9) is a linear classifier in $\ell^2$. $\phi$ maps vectors from $H_2$ into the feature space. We define a second mapping from $H_1$ to the feature space by $\omega(\boldsymbol{z}) := (e_1(\boldsymbol{z}), e_2(\boldsymbol{z}), \dots)$. For $\alpha = \sum_{i=1}^P \alpha_i \delta(\boldsymbol{z}^i)$, where $\delta(\boldsymbol{z}^i)$ is the Dirac delta, we recover the discrete classifier (6) and $\boldsymbol{w} = \sum_{i=1}^P \alpha_i \ \omega(\boldsymbol{z}^i)$. We observe that $\|f\|_{H_2}^2 = \boldsymbol{\alpha}^T \boldsymbol{K}^T \boldsymbol{K} \ \boldsymbol{\alpha} = \|\boldsymbol{X}^T \ \boldsymbol{w}\|_2^2$. A problem may arise if $\boldsymbol{z}^i$ belongs to a set of measure zero which does not obey the singular value decomposition of $k$. If this occurs $\delta(\boldsymbol{z}^i)$ may be set to the zero function.

Theorem 1 tells us that any measurement kernel $k$ applied to objects $\boldsymbol{x}$ and $\boldsymbol{z}$ can be expressed for almost all $\boldsymbol{x}$ and $\boldsymbol{z}$ as $k(\boldsymbol{x}, \boldsymbol{z}) = \langle\phi(\boldsymbol{x}), \omega(\boldsymbol{z})\rangle$, where $\langle.\rangle$ defines a dot product in some feature space for almost all $\boldsymbol{x}, \boldsymbol{z}$. Hence, we can define the a matrix $\boldsymbol{X} := (\phi(\boldsymbol{x}^1), \phi(\boldsymbol{x}^2), \dots, \phi(\boldsymbol{x}^L))$ of feature vectors for the $L$ column objects and a matrix $\boldsymbol{Z} := (\omega(\boldsymbol{z}^1), \omega(\boldsymbol{z}^2), \dots, \omega(\boldsymbol{z}^P))$ of feature vectors for the $P$ row objects and apply the results of Section 2.

## 4 Pairwise Data

An interesting special case occurs if row and column objects coincide. This kind of data is known as pairwise data [5, 4, 8] where the objects to be classified serve as features and vice versa. Like in Section 3 we can expand the measurement kernel via singular value decomposition but that would introduce two different mappings ($\phi$ and $\omega$) into the feature space. We will use one map for row and column objects and perform an eigenvalue decomposition. The consequence is that that eigenvalues may be negative (see the following theorem).

**Theorem 2 (Eigenvalue Expansion)** *Let definitions and assumptions be as in Theorem 1. Let $H_1 = H_2 = H$ and let $k$ be symmetric. Then there exists an expansion $k(\boldsymbol{x}, \boldsymbol{z}) = \sum_n \nu_n \, e_n(\boldsymbol{z}) \, e_n(\boldsymbol{x})$ which converges in the $L^2$-sense. The $\nu_n$ are the eigenvalues of $T_k$ with the corresponding orthonormal eigenfunctions $e_n$.*

**Corollary 2 (Minkowski Space Classification)** *Let the assumptions of Theorem 2 and $\int_H (k(\boldsymbol{x}, \boldsymbol{z}))^2 \, d\boldsymbol{z} \leq K^2$ for all $\boldsymbol{x}$ hold true. We define $\boldsymbol{w} := (\sqrt{|\nu_1|}\langle \alpha, e_1 \rangle_H, \sqrt{|\nu_2|}\langle \alpha, e_2 \rangle_H, \dots)$, $\phi(\boldsymbol{x}) := (\sqrt{|\nu_1|}e_1(\boldsymbol{x}), \sqrt{|\nu_2|}e_2(\boldsymbol{x}), \dots)$, and $\ell_S^2$ to denote $\ell^2$ with a given signature $S = (sign(\nu_1), sign(\nu_2), \dots)$.*

*Then the following holds true:*

$\|w\|_{\ell_S^2}^2 = \sum_n sign(\nu_n) \left( \sqrt{|\nu_n|} \, \langle \alpha, e_n \rangle_H \right)^2 = \sum_n \nu_n \langle \alpha, e_n \rangle_H^2 = \langle T_k \alpha, \alpha \rangle_H,$

$\|\phi(\boldsymbol{x})\|_{\ell_S^2}^2 = \sum_n \nu_n \, e_n(\boldsymbol{x})^2 = k(\boldsymbol{x}, \boldsymbol{x})$ *in the $L^2$ sense, and the following sum convergences absolutely and uniformly:*

$$f(\boldsymbol{x}) = \langle \boldsymbol{w}, \phi(\boldsymbol{x}) \rangle_{\ell_S^2} = \sum_n \nu_n \, \langle \alpha, e_n \rangle_H \, e_n(\boldsymbol{x}) \, . \tag{10}$$

Eq. (10) is a linear classifier in the Minkowski space $\ell_S^2$. For the discrete case $\alpha = \sum_{i=1}^P \alpha_i \delta(\boldsymbol{z}^i)$, the normal vector is $\boldsymbol{w} = \sum_{i=1}^P \alpha_i \phi(\boldsymbol{z}^i)$. In comparison to Corollary 1, we have $\|w\|_{\ell_S^2}^2 = \boldsymbol{\alpha}^T \boldsymbol{K} \, \boldsymbol{\alpha}$. and must assume that $\|\phi(\boldsymbol{x})\|_{\ell_S^2}^2$ does converge. Unfortunately, this can be assured in general only for almost all $\boldsymbol{x}$. If $k$ is both continuous and positive definite and if $H$ is compact, then the sum converges uniformly and absolutely for all $\boldsymbol{x}$ (Mercer).

## 5 Sparseness and Feature Selection

As mentioned in the text after optimization problem (4) $\boldsymbol{\alpha}$ may be not u nique and an additional regularization term is needed. We choose the regularization term such that it enforces sparseness and that it also can be used for feature selection. We choose "$\epsilon \|\boldsymbol{\alpha}\|_1$", where $\epsilon$ is the regularization parameter. We separate $\boldsymbol{\alpha}$ into a positive part $\boldsymbol{\alpha}^+$ and a negative part $\boldsymbol{\alpha}^-$ with $\boldsymbol{\alpha} = \boldsymbol{\alpha}^+ - \boldsymbol{\alpha}^-$ and $\alpha_i^+, \alpha_i^- \geq 0$ [11]. The dual optimization problem is then given by

$$\min_{\boldsymbol{\alpha}} \quad \frac{1}{2} (\boldsymbol{\alpha}^+ - \boldsymbol{\alpha}^-)^T \, \boldsymbol{K}^T \, \boldsymbol{K} \, (\boldsymbol{\alpha}^+ - \boldsymbol{\alpha}^-) - \tag{11}$$

$$\boldsymbol{1}^T \boldsymbol{Y} \, \boldsymbol{K} \, (\boldsymbol{\alpha}^+ - \boldsymbol{\alpha}^-) + \epsilon \, \boldsymbol{1}^T (\boldsymbol{\alpha}^+ + \boldsymbol{\alpha}^-)$$

$$\text{s.t.} \quad \boldsymbol{1}^T \boldsymbol{K} \, (\boldsymbol{\alpha}^+ - \boldsymbol{\alpha}^-) = 0 \, , \, C\boldsymbol{1} \geq \boldsymbol{\alpha}^+, \boldsymbol{\alpha}^- \geq \boldsymbol{0} \, .$$

If $\boldsymbol{\alpha}$ is sparse, i.e. if many $\alpha_i = \alpha_i^+ - \alpha_i^-$ are zero, the classification function $f(\boldsymbol{u}) = \langle \boldsymbol{w}, \boldsymbol{u} \rangle + b = \sum_{i=1}^P (\alpha_i^+ - \alpha_i^-) \, \langle \boldsymbol{z}^i, \boldsymbol{u} \rangle + b$ contains only few terms. This saves on the number of measurements $\langle \boldsymbol{z}^i, \boldsymbol{u} \rangle$ for new objects and yields to improved classification performance due to the reduced number of features $\boldsymbol{z}^i$ [15].

# 6 Application to DNA Microarray Data

We apply our new method to the DNA microarray data published in [9]. Column objects are samples from different brain tumors of the medullablastoma kind. The samples were obtained from 60 patients, which were treated in a similar way and the samples were labeled according to whether a patient responded well to chemo- or radiation therapy. Row objects correspond to genes. Transcriptions of 7,129 genes were tagged with fluorescent dyes and used as a probe in a binding assay. For every sample-gene pair, the fluorescence of the bound transcripts - a snapshot of the level of gene expression - was measured. This gave rise to a $60 \times 7,129$ real valued sample-gene matrix where each entry represents the level of gene expression in the corresponding sample. For more details see [9].

The task is now to construct a classifier which predicts therapy outcome on the basis of samples taken from new patients. The major problem of this classification task is the limited number of samples - given the large number of genes. Therefore, feature selection is a prerequisite for good generalization [6, 16]. We construct the classifier using a two step procedure. In a first step, we apply our new method on a $59 \times 7,129$ matrix, where one column object was withhold to avoid biased feature selection. We choose $\epsilon$ to be fairly large in order to obtain a sparse set of features. In a second step, we use the selected features only and apply our method once more on the reduced sample-gene matrix, but now with a small value of $\epsilon$. The $C$-parameter is used for regularization instead.

| Feature Selection / Classification | # F | # E | Feature Selection / Classification | $C$ | # F | # E |
|---|---|---|---|---|---|---|
| TrkC | 1 | 20 | P-SVM / C-SVM | 1.0 | 40/45/50 | 5/4/5 |
| statistic / SVM | | 15 | P-SVM / C-SVM | 0.01 | 40/45/50 | 5/5/5 |
| statistic / Comb1 | | 14 | P-SVM / P-SVM | 0.1 | 40/45/50 | 4/4/5 |
| statistic / KNN | 8 | 13 | | | | |
| statistic / Comb2 | | 12 | | | | |

Table 1: Benchmark results for DNA microarray data (for explanations see text). The table shows the classification error given by the number of wrong classifications ("E") for different numbers of selected features ("F") and for different values of the parameter $C$. The feature selection method is signal-to-noise-statistic and $t$-statitic denoted by "statistic" or our method P-SVM. Data are provided for "TrkC"-Gene classification, standard SVMs, weighted "TrkC"/SVM (Comb1), K nearest neighbor (KNN), combined SVM/TrkC/KNN (Comb2), and our procedure (P-SVM) used for classification. Except for our method (P-SVM), results were taken from [9].

Table 1 shows the result of a leave-one-out cross-validation procedure, where the classification error is given for different numbers of selected features. Our method (P-SVM) is compared with "TrkC"-Gene classification (one gene classification), standard SVMs, weighted "TrkC"/SVM-classification, K nearest neighbor (KNN), and a combined SVM/TrkC/KNN classifier. For the latter methods, feature selection was based on the correlation of features with classes using signal-to-noise-statistics and $t$-statistics [3]. For our method we use $C = 1.0$ and $0.1 \leq \epsilon \leq 1.5$ for feature selection in step one which gave rise to $10 - 1000$ selected features. The feature selection procedure (also a classifier) had its lowest misclassification rate between 20 and 40 features. For the construction of the classifier we used in step two $\epsilon = 0.01$. Our feature selection method clearly outperforms standard methods — the number of misclassification is down by a factor of 3 (for 45 selected genes).

**Acknowledgments**

We thank the anonymous reviewers for their hints to improve the paper. This work was funded by the DFG (SFB 618).

# References

[1] B. E. Boser, I. M. Guyon, and V. N. Vapnik. A training algorithm for optimal margin classifiers. In *Proc. of the 5th Annual ACM Workshop on Computational Learning Theory*, pages 144–152. ACM Press, Pittsburgh, PA, 1992.

[2] C. Cortes and V. N. Vapnik. Support vector networks. *Machine Learning*, 20:273–297, 1995.

[3] R. Golub, D. K. Slonim, P. Tamayo, C. Huard, M. Gaasenbeek, J. P. Mesirov, H. Coller, M. Loh, J. R. Downing, M. A. Caligiuri, C. D. Bloomfield, and E. S. Lander. Molecular classification of cancer: Class discovery and class prediction by gene expression monitoring. *Science*, 286(5439):531–537, 1999.

[4] T. Graepel, R. Herbrich, P. Bollmann-Sdorra, and K. Obermayer. Classification on pairwise proximity data. In *NIPS 11*, pages 438–444, 1999.

[5] T. Graepel, R. Herbrich, B. Schölkopf, A. J. Smola, P. L. Bartlett, K.-R. Müller, K. Obermayer, and R. C. Williamson. Classification on proximity data with LP–machines. In *ICANN 99*, pages 304–309, 1999.

[6] I. Guyon, J. Weston, S. Barnhill, and V. Vapnik. Gene selection for cancer classification using support vector machines. *Mach. Learn.*, 46:389–422, 2002.

[7] S. Hochreiter and K. Obermayer. Classification of pairwise proximity data with support vectors. In *The Learning Workshop*. Y. LeCun and Y. Bengio, 2002.

[8] T. Hofmann and J. Buhmann. Pairwise data clustering by deterministic annealing. *IEEE Trans. on Pat. Analysis and Mach. Intell.*, 19(1):1–14, 1997.

[9] S. L. Pomeroy, P. Tamayo, M. Gaasenbeek, L. M. Sturla, M. Angelo, M. E. McLaughlin, J. Y. H. Kim, L. C. Goumnerova, P. M. Black, C. Lau, J. C. Allen, D. Zagzag, J. M. Olson, T. Curran, C. Wetmore, J. A. Biegel, T. Poggio, S. Mukherjee, R. Rifkin, A. Califano, G. Stolovitzky, D. N. Louis, J. P. Mesirov, E. S. Lander, and T. R. Golub. Prediction of central nervous system embryonal tumour outcome based on gene expression. *Nature*, 415(6870):436–442, 2002.

[10] V. Roth, J. Buhmann, and J. Laub. Pairwise clustering is equivalent to classical k-means. In *The Learning Workshop*. Y. LeCun and Y. Bengio, 2002.

[11] B. Schölkopf and A. J. Smola. *Learning with kernels — Support Vector Machines, Reglarization, Optimization, and Beyond*. MIT Press, Cambridge, 2002.

[12] J. Shawe-Taylor, P. L. Bartlett, R. C. Williamson, and M. Anhtony. A framework for structural risk minimisation. In *Comp. Learn. Th.*, pages 68–76, 1996.

[13] J. Shawe-Taylor, P. L. Bartlett, R. C. Williamson, and M. Anhtony. Structural risk minimization over data-dependent hierarchies. *IEEE Transactions on Information Theory*, 44:1926–1940, 1998.

[14] J. Shawe-Taylor and N. Cristianini. On the generalisation of soft margin algorithms. Technical Report NC2-TR-2000-082, NeuroCOLT2, Department of Computer Science, Royal Holloway, University of London, 2000.

[15] V. Vapnik. *The nature of statistical learning theory*. Springer, NY, 1995.

[16] J. Weston, S. Mukherjee, O. Chapelle, M. Pontil, T. Poggio, and V. Vapnik. Feature selection for SVMs. In *NIPS 12*, pages 668–674, 2000.
